# Probabilistic Interpretation of Population Codes

Richard S. Zemel          Peter Dayan          Alexandre Pouget
zemel@u.arizona.edu    dayan@ai.mit.edu    alex@salk.edu

## Abstract

We present a theoretical framework for population codes which generalizes naturally to the important case where the population provides information about a whole probability distribution over an underlying quantity rather than just a single value. We use the framework to analyze two existing models, and to suggest and evaluate a third model for encoding such probability distributions.

## 1 Introduction

Population codes, where information is represented in the activities of whole populations of units, are ubiquitous in the brain. There has been substantial work on how animals should and/or actually do extract information about the underlying encoded quantity.[5,3,11,9,12] With the exception of Anderson,[1] this work has concentrated on the case of extracting a *single value* for this quantity. We study ways of characterizing the joint activity of a population as coding a whole *probability distribution* over the underlying quantity.

Two examples motivate this paper: place cells in the hippocampus of freely moving rats that fire when the animal is at a particular part of an environment,[8] and cells in area MT of monkeys firing to a random moving dot stimulus.[7] Treating the activity of such populations of cells as reporting a single value of their underlying variables is inadequate if there is (a) insufficient information to be sure (*eg* if a rat can be uncertain as to whether it is in place $x_A$ or $x_B$ then perhaps place cells for both locations should fire; or (b) if multiple values underlie the input, as in the whole distribution of moving random dots in the motion display. Our aim is to capture the computational power of representing a probability distribution over the underlying parameters.[6]

RSZ is at University of Arizona, Tucson, AZ 85721; PD is at MIT, Cambridge, MA 02139; AP is at Georgetown University, Washington, DC 20007. This work was funded by McDonnell-Pew, NIH, AFOSR and startup funds from all three institutions.

In this paper, we provide a general statistical framework for population codes, use it to understand existing methods for coding probability distributions and also to generate a novel method. We evaluate the methods on some example tasks.

## 2   Population Code Interpretations

The starting point for almost all work on neural population codes is the neurophysiological finding that many neurons respond to particular variable(s) underlying a stimulus according to a unimodal tuning function such as a Gaussian. This characterizes cells near the sensory periphery and also cells that report the results of more complex processing, including receiving information from groups of cells that themselves have these tuning properties (in MT, for instance). Following Zemel & Hinton's[13] analysis, we distinguish two spaces: the *explicit* space which consists of the activities $\mathbf{r} = \{r_i\}$ of the cells in the population, and a (typically low dimensional) *implicit* space which contains the underlying information $\mathcal{X}$ that the population encodes in which they are tuned. All processing on the basis of the activities $\mathbf{r}$ has to be referred to the implicit space, but it itself plays no explicit role in determining activities.

Figure 1 illustrates our framework. At the top is the measured activities of a population of cells. There are two key operations. **Encoding:** What is the relationship between the activities $\mathbf{r}$ of the cells and the underlying quantity in the world $\mathcal{X}$ that is represented? **Decoding:** What information about the quantity $\mathcal{X}$ can be extracted from the activities? Since neurons are generally noisy, it is often convenient to characterize encoding (operations A and B) in a probabilistic way, by specifying $\mathcal{P}[\mathbf{r}|\mathcal{X}]$. The simplest models make a further assumption of conditional independence of the different units given the underlying quantity $\mathcal{P}[\mathbf{r}|\mathcal{X}] = \prod_i \mathcal{P}[r_i|\mathcal{X}]$ although others characterize the degree of correlation between the units. If the encoding model is true, then a Bayesian decoding model specifies that the information $\mathbf{r}$ carries about $\mathcal{X}$ can be characterized precisely as: $\mathcal{P}[\mathcal{X}|\mathbf{r}] \propto \mathcal{P}[\mathbf{r}|\mathcal{X}]\mathcal{P}[\mathcal{X}]$, where $\mathcal{P}[\mathcal{X}]$ is the prior distribution about $\mathcal{X}$ and the constant of proportionality is set so that $\int_{\mathcal{X}} \mathcal{P}[\mathcal{X}|\mathbf{r}]d\mathcal{X} = 1$. Note that starting with a deterministic quantity $\mathcal{X}$ in the world, encoding in the firing rates $\mathbf{r}$, and decoding it (operation C) results in a probability distribution over $\mathcal{X}$. This uncertainty arises from the stochasticity represented by $\mathcal{P}[\mathbf{r}|\mathcal{X}]$. Given a loss function, we could then go on to extract a single value from this distribution (operation D).

We attack the common assumption that $\mathcal{X}$ is a single value of some variable $\mathbf{x}$, *eg* the single position of a rat in an environment, or the single coherent direction of motion of a set of dots in a direction discrimination task. This does not capture the subtleties of certain experiments, such as those in which rats can be made to be uncertain about their position, or in which one direction of motion predominates yet there are several simultaneous motion directions.[7] Here, the natural characterization of $\mathcal{X}$ is actually a whole probability distribution $\mathcal{P}[\mathbf{x}|\omega]$ over the value of the variable $\mathbf{x}$ (perhaps plus extra information about the number of dots), where $\omega$ represents all the available information. We can now cast two existing classes of proposals for population codes in terms of this framework.

**The Poisson Model**

Under the Poisson encoding model, the quantity $\mathcal{X}$ encoded is indeed one particular value which we will call $\mathbf{x}$, and the activities of the individual units are independent,

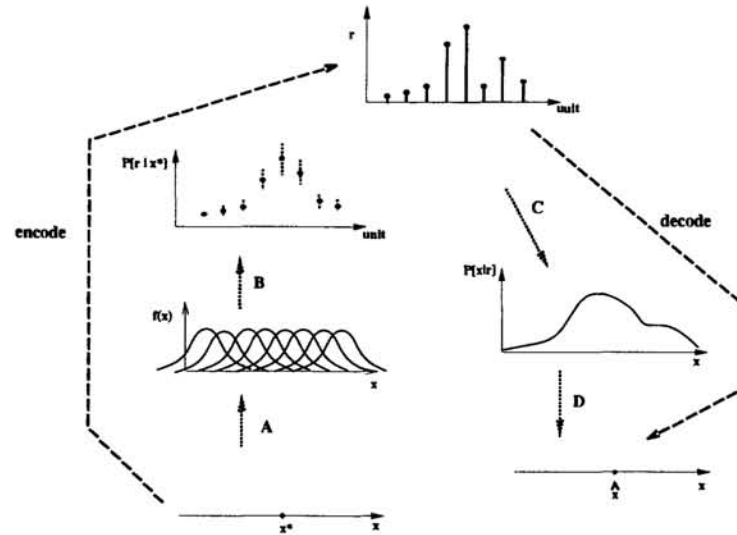

Figure 1: *Left:* encoding maps $\mathcal{X}$ from the world through tuning functions (A) into mean activities (B), leading to *Top:* observed activities r. We assume complete knowledge of the variables governing systematic changes to the activities of the cells. Here $\mathcal{X}$ is a single value $\mathbf{x}^*$ in the space of underlying variables. *Right:* decoding extracts $\mathcal{P}[\mathcal{X}|\mathbf{r}]$ (C); a single value can be picked (D) from this distribution given a loss function.

---

with the terms $\mathcal{P}[r_i|\mathbf{x}] = e^{-f_i(\mathbf{x})}(f_i(\mathbf{x}))^{r_i}/r_i!$. The activity $r_i$ could, for example, be the number of spikes the cell emits in a fixed time interval following the stimulus onset. A typical form for the tuning function $f_i(\mathbf{x})$ is Gaussian $f_i(\mathbf{x}) \propto e^{-(\mathbf{x}-\mathbf{x}_i)^2/2\sigma^2}$ about a preferred value $\mathbf{x}_i$ for cell $i$. The Poisson decoding model is:[3,11,9,12]

$$\log \mathcal{P}[\mathbf{x}|\mathbf{r}] = \mathcal{K} - \sum_i f_i(\mathbf{x}) + \sum_i r_i \log f_i(\mathbf{x}) \tag{1}$$

where $\mathcal{K}$ is a constant with respect to $\mathbf{x}$.

Although simple, the Poisson model makes the the assumption criticized above, that $\mathcal{X}$ is just a single value $\mathbf{x}$. We argued for a characterization of the quantity $\mathcal{X}$ in the world that the activities of the cells encode as now $\mathcal{P}[\mathbf{x}|\omega]$. We describe below a method of encoding that takes exactly this definition of $\mathcal{X}$. However, wouldn't $\mathcal{P}[\mathbf{x}|\mathbf{r}]$ from Equation 1 be good enough? Not if $f_i(\mathbf{x})$ are Gaussian, since

$$\log \mathcal{P}[\mathbf{x}|\mathbf{r}] = \mathcal{K}' - \frac{1}{2}\left(\frac{\sum_i r_i}{\sigma^2}\right)\left(\mathbf{x} - \frac{\sum_i r_i \mathbf{x}_i}{\sum_i r_i}\right)^2,$$

completing the square, implying that $\mathcal{P}[\mathbf{x}|\mathbf{r}]$ is Gaussian, and therefore inevitably unimodal. Worse, the width of this distribution goes down with $\sum_i r_i$, making it, in most practical cases, a close approximation to a delta function.

## The KDE Model

Anderson[1,2] set out to represent whole probability distributions over $\mathbf{x}$ rather than just single values. Activities $\mathbf{r}$ represent distribution $\hat{\mathcal{P}}^{\mathbf{r}}(\mathbf{x})$ through a linear combination of basis functions $\psi_i(\mathbf{x})$, *ie* $\hat{\mathcal{P}}^{\mathbf{r}}(\mathbf{x}) = \sum_i r_i' \psi_i(\mathbf{x})$ where $r_i'$ are normalized such that $\hat{\mathcal{P}}^{\mathbf{r}}(\mathbf{x})$ is a probability distribution. The kernel functions $\psi_i(\mathbf{x})$ are *not*

the tuning functions $f_i(\mathbf{x})$ of the cells that would commonly be measured in an experiment. They need have *no* neural instantiation; instead, they form part of the interpretive structure for the population code. If the $\psi_i(\mathbf{x})$ are probability distributions, and so are positive, then the range of spatial frequencies in $\mathcal{P}[\mathbf{x}|\omega]$ that they can reproduce in $\hat{\mathcal{P}}^\mathbf{r}(\mathbf{x})$ is likely to be severely limited.

In terms of our framework, the KDE model specifies the method of decoding, and makes encoding its corollary. Evaluating KDE requires some choice of encoding – representing $\mathcal{P}[\mathbf{x}|\omega]$ by $\hat{\mathcal{P}}^\mathbf{r}(\mathbf{x})$ through appropriate $\mathbf{r}$. One way to encode is to use the Kullback-Leibler divergence as a measure of the discrepancy between $\mathcal{P}[\mathbf{x}|\omega]$ and $\sum_i r'_i \psi_i(\mathbf{x})$ and use the expectation-maximization (EM) algorithm to fit the $\{r'_i\}$, treating them as mixing proportions in a mixture model.[4] This relies on $\{\psi_i(\mathbf{x})\}$ being probability distributions themselves. The *projection method*[1] is a one-shot linear filtering based alternative using the $\mathcal{L}_2$ distance. $r_i$ are computed as a projection of $\mathcal{P}[\mathbf{x}|\omega]$ onto tuning functions $f_i(\mathbf{x})$ that are calculated from $\psi_j(\mathbf{x})$.

$$r_i = \int_\mathbf{x} \mathcal{P}[\mathbf{x}|\omega] f_i(\mathbf{x}) d\mathbf{x} \quad \mathbf{f_i}(\mathbf{x}) = \sum_j \mathbf{A}_{ij}^{-1} \psi_j(\mathbf{x}) \quad \mathbf{A}_{ij} = \int_\mathbf{x} \psi_i(\mathbf{x}) \psi_j(\mathbf{x}) d\mathbf{x} \qquad (2)$$

$f_i(\mathbf{x})$ are likely to need regularizing,[1] particularly if the $\psi_i(\mathbf{x})$ overlap substantially.

## 3  The Extended Poisson Model

The KDE model is likely to have difficulty capturing in $\hat{\mathcal{P}}^\mathbf{r}(\mathbf{x})$ probability distributions $\mathcal{P}[\mathbf{x}|\omega]$ that include high frequencies, such as delta functions. Conversely, the standard Poisson model decodes almost *any* pattern of activities $\mathbf{r}$ into something that rapidly approaches a delta function as the activities increase. Is there any middle ground?

We extend the standard Poisson encoding model to allow the recorded activities $\mathbf{r}$ to depend on general $\mathcal{P}[\mathbf{x}|\omega]$, having Poisson statistics with mean:

$$\langle r_i \rangle = \int_\mathbf{x} \mathcal{P}[\mathbf{x}|\omega] f_i(\mathbf{x}) d\mathbf{x}. \qquad (3)$$

This equation is identical to that for the KDE model (Equation 2), except that variability is built into the Poisson statistics, and decoding is now required to be the Bayesian inverse of encoding. Note that since $r_i$ depends stochastically on $\mathcal{P}[\mathbf{x}|\omega]$, the full Bayesian inverse will specify a distribution $\mathcal{P}[\mathcal{P}[\mathbf{x}|\omega]|\mathbf{r}]$ over possible distributions. We summarize this by an approximation to its most likely member— we perform an approximate form of maximum likelihood, not in the value of $\mathbf{x}$, but in distributions over $\mathbf{x}$. We approximate $\mathcal{P}[\mathbf{x}|\omega]$ as a piece-wise constant histogram which takes the value $\phi_j$ in $(\mathbf{x}_j, \mathbf{x}_{j+1}]$, and $f_i(\mathbf{x})$ by a piece-wise constant histogram that take the values $f_{ij}$ in $(\mathbf{x}_j, \mathbf{x}_{j+1}]$. Generally, the maximum *a posteriori* estimate for $\{\phi_j\}$ can be shown to be derived by maximizing:

$$L(\{\hat{\phi}_j\}) = \sum_i r_i \log \left[ \sum_j \hat{\phi}_j f_{ij} \right] - \epsilon \sum_j \left( \hat{\phi}_j - \hat{\phi}_{j+1} \right)^2 \qquad (4)$$

where $\epsilon$ is the variance of a smoothness prior. We use a form of EM to maximize the likelihood and adopt the crude approximation of averaging neighboring values

| Operation | Extended Poisson | KDE (Projection) | KDE (EM) |
|---|---|---|---|
| Encode $\langle r_i \rangle$ | $\langle r_i \rangle = h\left[\int_x P[x|\omega]f_i(x)dx\right]$ <br> $f_i(x) = R_{max}\mathcal{N}(x_i, \sigma)$ | $\langle r_i \rangle = h\left[R_{max}\int_x P[x|\omega]f_i(x)dx\right]$ <br> $f_i(x) = \sum_j A_{ij}^{-1}\psi_j(x)$ <br> $A_{ij} = \int_x \psi_i(x)\psi_j(x)dx$ | $\langle r_i \rangle = h\left[R_{max}r_i'\right]$ <br> $r_i'$ to max. $L$ |
| Decode $\hat{\mathcal{P}}^r(x)$ | $\hat{\mathcal{P}}^r(x)$ to max. $L$ <br> $\hat{r}_i = \int_x \hat{\mathcal{P}}^r(x)f_i(x)dx \approx \sum_j \phi_j f_{ij}$ | $\hat{\mathcal{P}}^r(x) = \sum_i r_i'\psi_i(x)$ <br> $r_i' = r_i/\sum_j r_j$ | $\hat{\mathcal{P}}^r(x) = \sum_i r_i'\psi_i(x)$ |
| Likelihood | $L = \log \mathcal{P}[\{\phi_j\}|\{r_i\}] \approx \sum_i r_i \log \hat{r}_i$ | | $L = \int_x P[x|\omega]\log \hat{\mathcal{P}}^r(x)dx$ |
| Error | $G = \sum_i r_i \log(r_i/\hat{r}_i)$ | $E = \int_x \left[\hat{\mathcal{P}}^r(x) - P[x|\omega]\right]^2 dx$ | $G = \int_x P[x|\omega]\log\frac{P[x|\omega]}{\hat{\mathcal{P}}^r(x)}dx$ |

Table 1: A summary of the key operations with respect to the framework of the interpretation methods compared here. $h[]$ is a rounding operator to ensure integer firing rates, and $\psi_i(x) = \mathcal{N}(x_i, \sigma)$ are the kernel functions for the KDE method.

of $\hat{\phi}_j$ on successive iterations. By comparison with the linear decoding of the KDE method, Equation 4 offers a *non-linear* way of combining a set of activities $\{r_i\}$ to give a probability distribution $\hat{\mathcal{P}}^r(x)$ over the underlying variable x. The computational complexities of Equation 4 are irrelevant, since decoding is only an implicit operation that the system need never actually perform.

## 4   Comparing the Models

We illustrate the various models by showing the faithfulness with which they can represent two bimodal distributions. We used $\sigma = 0.3$ for the kernel functions (KDE) and the tuning functions (extended Poisson model) and used 50 units whose $x_i$ were spaced evenly in the range $x = [-10, 10]$. Table 1 summarizes the three methods.

Figure 2a shows the decoded version of a mixture of two broad Gaussians $1/2\mathcal{N}[-2, 1] + 1/2\mathcal{N}[2, 1]$. Figure 2b shows the same for a mixture of two narrow Gaussians $\frac{1}{2}\mathcal{N}[-2, .2] + \frac{1}{2}\mathcal{N}[2, .2]$. All the models work well for representing the broad Gaussians; both forms of the KDE model have difficulty with the narrow Gaussians. The EM version of KDE puts all its weight on the nearest kernel functions, and so is too broad; the projection version 'rings' in its attempt to represent the narrow components of the distributions. The extended Poisson model reconstructs with greater fidelity.

## 5   Discussion

Informally, we have examined the consequences of the seemingly obvious step of saying that if a rat, for instance, is uncertain about whether it is at one of two places, then place cells representing both places could be activated. The complications

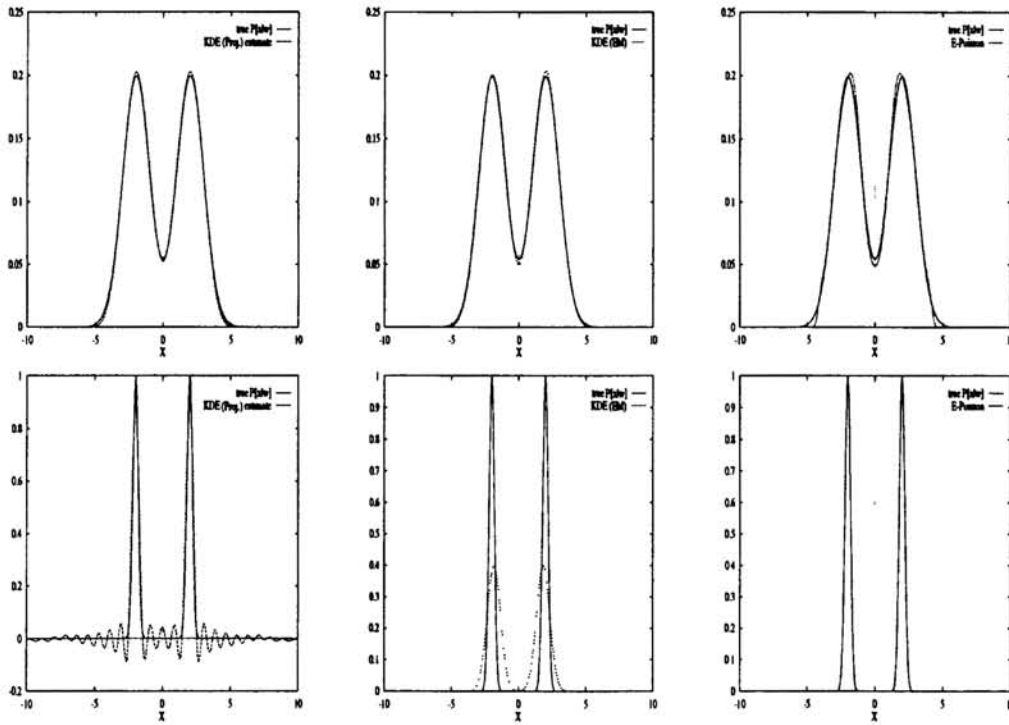

Figure 2: a) (upper) All three methods provide a good fit to the bimodal Gaussian distribution when its variance is sufficiently large ($\tau = 1.0$). b) (lower) The KDE model has difficulty when $\tau = 0.2$.

come because the structure of the interpretation changes – for instance, one can no longer think of maximum likelihood methods to extract a single value from the code directly.

One main fruit of our resulting framework is a method for encoding and decoding probability distributions that is the natural extension of the (provably inadequate) standard Poisson model for encoding and decoding single values. Cells have Poisson statistics about a mean determined by the integral of the whole probability distribution, weighted by the tuning function of the cell. We suggested a particular decoding model, based on an approximation to maximum likelihood decoding to a discretized version of the whole probability distribution, and showed that it reconstructs broad, narrow and multimodal distributions more accurately than either the standard Poisson model or the kernel density model. Stochasticity is built into our method, since the units are supposed to have Poisson statistics, and it is therefore also quite robust to noise. The decoding method is not biologically plausible, but provides a quantitative lower bound to the faithfulness with which a set of activities can code a distribution.

Stages of processing subsequent to a population code might either *extract* a single value from it to control behavior, or *integrate* it with information represented in other population codes to form a combined population code. Both operations must be performed through standard neural operations such as taking non-linear weighted sums and possibly products of the activities. We are interested in how much information is preserved by such operations, as measured against the non-biological

standard of our decoding method. Modeling extraction requires modeling the loss function – there is some empirical evidence about this from a motion experiment in which electrical stimulation of MT cells was pitted against input from a moving stimulus.[10] However, much works remains to be done.

Integrating two or more population codes to generate the output in the form of another population code was stressed by Hinton,[6] who noted that it directly relates to the notion of generalized Hough transforms. We are presently studying how a system can *learn* to perform this combination, using the EM-based decoder to generate targets. One special concern for combination is how to understand noise. For instance, the visual system can be behaviorally extraordinarily sensitive – detecting just a handful of photons. However, the outputs of real cells at various stages in the system are apparently quite noisy, with Poisson statistics. If noise is added at every stage of processing and combination, then the final population code will not be very faithful to the input. There is much current research on the issue of the creation and elimination of noise in cortical synapses and neurons.

A last issue that we have not treated here is certainty or magnitude. Hinton's[6] idea of using the sum total activity of a population to code the certainty in the existence of the quantity they represent is attractive, provided that there is some independent way of knowing what the scale is for this total. We have used this scaling idea in both the KDE and the extended Poisson models. In fact, we can go one stage further, and interpret greater activity still as representing information about the existence of multiple objects or multiple motions. However, this treatment seems less appropriate for the place cell system — the rat is presumably always certain that it is somewhere. There it is plausible that the absolute level of activity could be coding something different, such as the familiarity of a location.

An entire collection of cells is a terrible thing to waste on representing just a single value of some quantity. Representing a whole probability distribution, at least with some fidelity, is not more difficult, provided that the interpretation of the encoding and decoding are clear. We suggest some steps in this direction.

# References

[1] Anderson, CH (1994). *International Journal of Modern Physics C*, **5**, 135–137.

[2] Anderson, CH & Van Essen, DC (1994). In *Computational Intelligence Imitating Life*, 213–222. New York: IEEE Press.

[3] Baldi, P & Heiligenberg, W (1988). *Biological Cybernetics*, **59**, 313-318.

[4] Dempster, AP, Laird, NM & Rubin, DB (1997). *Proceedings of the Royal Statistical Society*, B **39**, 1-38.

[5] Georgopoulos, AP, Schwartz, AB & Kettner, RE (1986). *Science*, **243**, 1416–1419.

[6] Hinton, GE (1992). *Scientific American*, **267(3)** 105-109.

[7] Newsome, WT, Britten, KH & Movshon, JA (1989). *Nature*, **341**, 52-54.

[8] O'Keefe, J & Dostrovsky, J (1971). *Brain Research*, **34**, 171-175.

[9] Salinas, E & Abbott, LF (1994). *Journal of Computational Neuroscience*, **1**, 89–107.

[10] Salzman, CD & Newsome, WT (1994). *Science*, **264**, 231-237.

[11] Seung, HS & Sompolinsky, H (1993). *Proceedings of the National Academy of Sciences, USA*, **90**, 10749–10753.

[12] Snippe, HP (1996). *Neural Computation*, **8**, 29–37.

[13] Zemel, RS & Hinton, GE (1995). *Neural Computation*, **7**, 549–564.
